# Generalization Abilities of Cascade Network Architectures

**E. Littmann**[*]
Department of Information Science
Bielefeld University
D-4800 Bielefeld, FRG
littmann@techfak.uni-bielefeld.de

**H. Ritter**
Department of Information Science
Bielefeld University
D-4800 Bielefeld, FRG
helge@techfak.uni-bielefeld.de

## Abstract

In [5], a new incremental cascade network architecture has been presented. This paper discusses the properties of such cascade networks and investigates their generalization abilities under the particular constraint of small data sets. The evaluation is done for cascade networks consisting of local linear maps using the Mackey-Glass time series prediction task as a benchmark. Our results indicate that to bring the potential of large networks to bear on the problem of *extracting information from small data sets without running the risk of overfitting*, deeply cascaded network architectures are more favorable than shallow broad architectures that contain the same number of nodes.

## 1 Introduction

For many real-world applications, a major constraint for the successful learning from examples is the limited number of examples available. Thus, methods are required, that can learn from small data sets. This constraint makes the problem of generalization particularly hard. If the number of adjustable parameters in a

---

[*]to whom correspondence should be sent

network approaches the number of training examples, the problem of *overfitting* occurs and generalization becomes very poor. This severely limits the size of networks applicable to a learning task with a small data set. To achieve good generalization also in these cases, particular attention must be paid to *a proper architecture* chosen for the network. The better the architecture matches the structure of the problem at hand, the better is the chance to achieve good results even with small data sets and small numbers of units.

In the present paper, we address this issue for the class of so called *Cascade Network Architectures* [5, 6] on the basis of an empirical approach, where we use the Mackey-Glass time series prediction as a benchmark problem. In our experiments we want to exploit the potential of large networks to bear on the problem of *extracting information from small data sets without running the risk of overfitting*. Our results indicate that it is more favorable to use deeply cascaded network architectures than shallow broad architectures, provided the same number of nodes is used in both cases. The width of each individual layer is essentially determined by the size of the training data set. The cascade depth is then matched to the total number of nodes available.

## 2   Cascade Architecture

So far, mainly architectures with few layers containing many units have been considered, while there has been very little research on narrow, but deeply cascaded networks. One of the few exceptions is the work of Fahlman [1], who proposed networks trained by the cascade-correlation algorithm. In his original approach, training is strictly feed-forward and the nonlinearity is achieved by incrementally adding perceptron units trained to maximize the covariance with the residual error.

### 2.1   Construction Algorithm

In [5] we presented a new incremental cascade network architecture based on *error minimization* instead of *covariance maximization*. This leads to an architecture that differs significantly from Fahlman's proposal and allows an *inversion of the construction process* of the network. Thus, at each stage of the construction of the network *all* cascaded modules provide an approximation of the target function $t(\xi)$, albeit corresponding to different states of convergence (Fig. 1).

The algorithm starts with the training of a neural module with output $\mathbf{y}^{(0)}$ to approximate a target function $\mathbf{t}(\xi)$, yielding

$$\mathbf{y}^{(0)}(\xi) = f^{(0)}\left(\mathbf{w}^{(0)}, \mathbf{x}^{(0)}(\xi)\right), \tag{1}$$

the superscript $^{(0)}$ indicating the cascade level. After an arbitrary number of training epochs, the weight vector $\mathbf{w}^{(0)}$ becomes "frozen". Now we add the output $\mathbf{y}^{(0)}$ of this module as a *virtual* input unit and train another neural module *as new output*

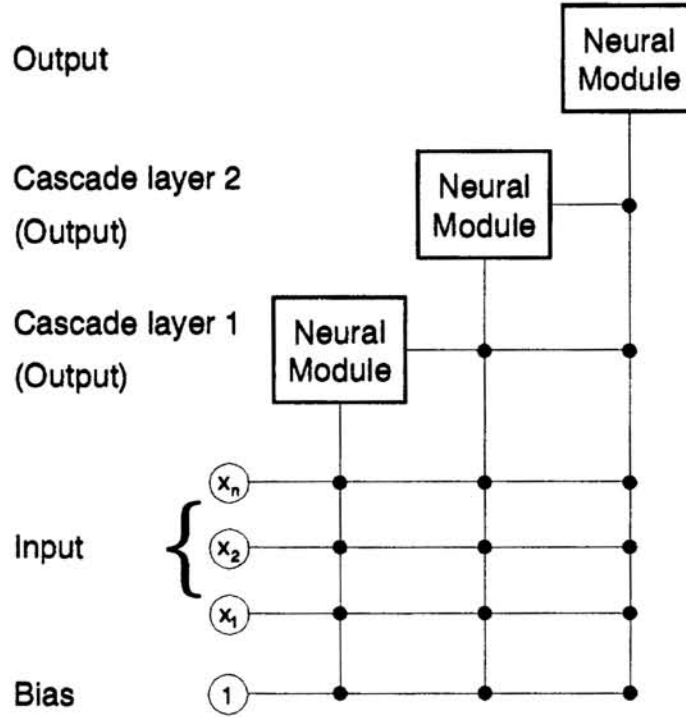

Figure 1: Cascade Network Architecture

*unit* $y^{(1)}$ with

$$y^{(1)} = f^{(1)}\left(\mathbf{w}^{(1)}, \mathbf{x}^{(1)}(\xi)\right). \tag{2}$$

where $\mathbf{x}^{(1)}(\xi) = \{\mathbf{x}^{(0)}(\xi), \mathbf{y}^{(0)}(\xi)\}$ denotes the extended input. This procedure can be iterated arbitrarily and generates a network structure as shown in Fig. 1.

## 2.2   Cascade Modules

The details and advantages of this approach are discussed in [5, 6]. In particular, this architecture can be applied to *any arbitrary nonlinear* module. It *does not* rely on the availability of a procedure for error backpropagation. Therefore, it is also applicable to (and has been extensively tested with) pure feed-forward approaches like simple perceptrons [5] and vector quantization or "Local linear maps" ("LLM networks") [6, 7].

## 2.3   Local Linear Maps

LLM networks have been introduced earlier ((Fig. 2); for details, cf. [11, 12]) and are related to the GRBF–approach [10] and the self–organizing maps [2, 3, 11]. They consist of $N$ units $r = 1, \ldots, N$, with an input weight vector $\mathbf{w}_r^{(in)} \in \mathbb{R}^L$, an output weight vector $\mathbf{w}_r^{(out)} \in \mathbb{R}^M$ and a $MxL$-matrix $\mathbf{A}_r$ for each unit $r$.

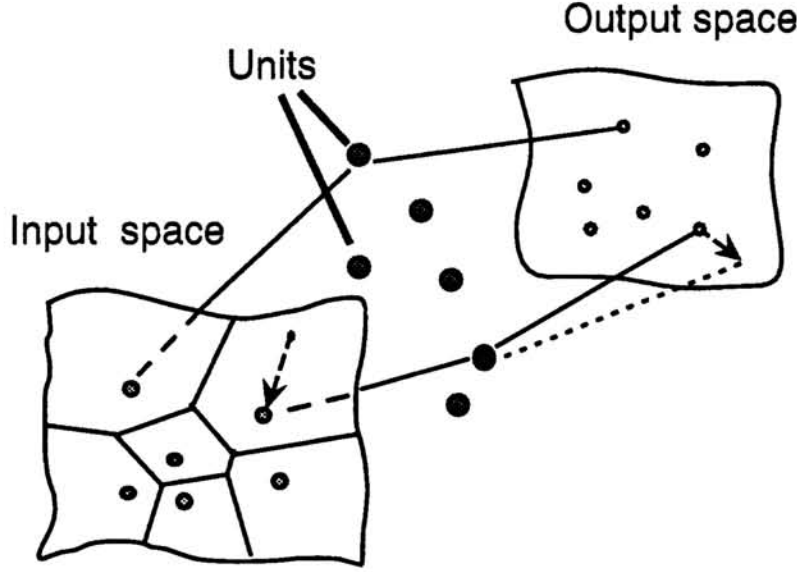

Figure 2: LLM Network Architecture

The output $\mathbf{y}^{(net)}$ of a single LLM-network for an input feature vector $\mathbf{x} \in \mathbb{R}^L$ is

$$\mathbf{y}^{(net)}(\mathbf{x}) = \mathbf{y}_s(\mathbf{x}) = \mathbf{w}_s^{(out)} + \mathbf{A}_s(\mathbf{x} - \mathbf{w}_s^{(in)}), \tag{3}$$

the "winner" node $s$ determined by the minimality condition

$$\|\mathbf{x} - \mathbf{x}_s\| = min_r\|\mathbf{x} - \mathbf{w}_r^{(in)}\|. \tag{4}$$

This leads to the learning steps for a training sample $(\mathbf{x}^{(\alpha)}, \mathbf{y}^{(\alpha)})$:

$$\Delta\mathbf{w}_s^{(in)} \quad = \quad \epsilon_1(\mathbf{x}^{(\alpha)} - \mathbf{w}_s^{(in)}), \tag{5}$$

$$\Delta\mathbf{w}_s^{(out)} \quad = \quad \epsilon_2(\mathbf{y}^{(\alpha)} - \mathbf{w}_s^{(out)}) - \mathbf{A}_s\Delta\mathbf{w}_s^{(in)}, \text{ and} \tag{6}$$

$$\Delta\mathbf{A}_s \quad = \quad \epsilon_3(d_s^2)^{-1}(\mathbf{y}^{(\alpha)} - \mathbf{y}^{(net)})(\mathbf{x}^{(\alpha)} - \mathbf{w}_s^{(in)})^T, \tag{7}$$

applied for $T$ samples $(\mathbf{x}^{(\alpha)}, \mathbf{y}^{(\alpha)}), \alpha = 1, 2, \ldots T$, and $0 < \epsilon_i << 1$, $i = 1, 2, 3$ denote learning step sizes. The additional term in (6), not given in [11, 12], leads to a better decoupling of the effects of (5) and (6,7).

## 3  Experiments

In order to evaluate the generalization performance of this architecture, we consider the problem of time series prediction based on the Mackey–Glass differential equation, for which results of other networks already have been reported in the literature.

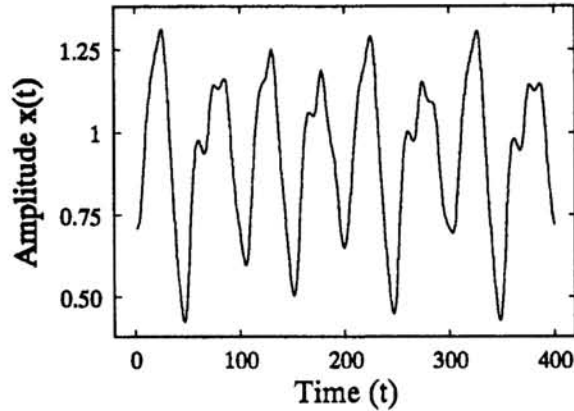

Figure 3: Mackey–Glass function

## 3.1   Time Series Prediction

Lapedes and Farber [4] introduced the prediction of chaotic time series as a benchmark problem. The data is based on the Mackey-Glass differential equation [8]:

$$\dot{x}(t) = -bx(t) + (ax(t - \tau))/(1 + x^{10}(t - \tau)). \tag{8}$$

With the parameters $a = 0.2$, $b = 0.1$, and $\tau = 17$, this equation produces a chaotic time series with a strange attractor of fractal dimension $d \approx 2.1$ (Fig. 3). The input data is a vector $\mathbf{x}(t) = \{x(t), x(t - \Delta), x(t - 2\Delta), x(t - 3\Delta)\}^T$. The learning task is defined to predict the value $x(t + P)$. To facilitate comparison, we adopt the standard choice $\Delta = 6$ and $P = 85$. Results with these parameters have been reported in [4, 9, 13].

The data was generated by integration with 30 steps per time unit. We performed different numbers of training epochs with samples randomly chosen from training sets consisting of 500 (5000 resp.) samples. The performance was measured on an independent test set of 5000 samples. All results are averages over ten runs. The error measure is the *normalized root mean square error* (**NRMSE**), i.e. predicting the average value yields an error value of 1.

## 4   Results and Discussion

The training of the single LLM networks was performed without extensive parameter tuning. If fine tuning for each cascade unit would be necessary, the training would be unattractively expensive.

The first results were achieved with cascade networks consisting of LLM units after 30 training epochs per layer on a learning set of 500 samples. Figs. 4 and 5 represent the performance of such LLM cascade networks *on the independent test set* for different numbers of cascaded layers as a function of the number of nodes per layer

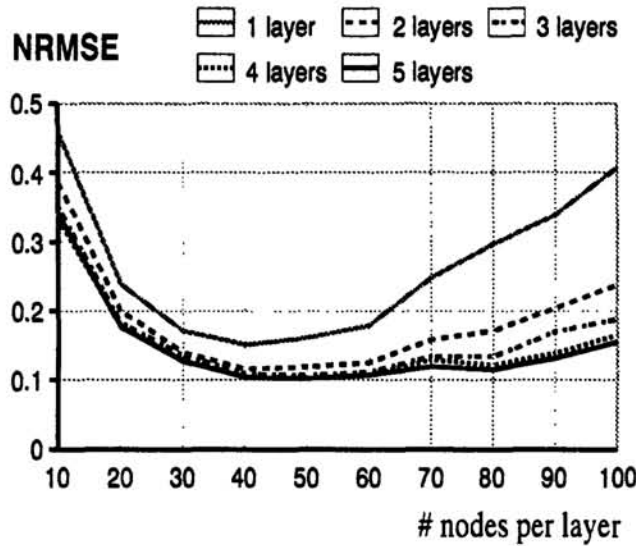

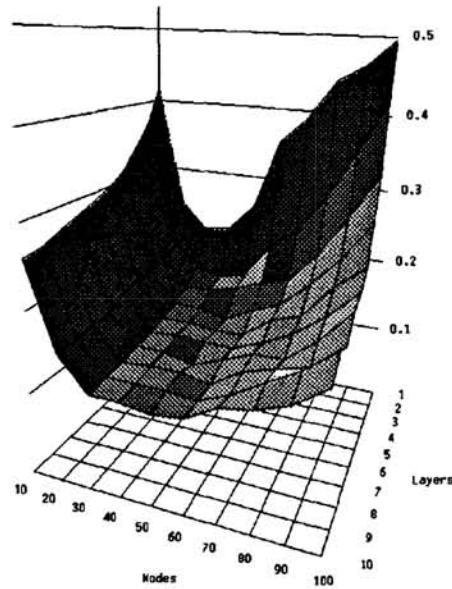

Figure 4: Iso-Layer-Dependence            Figure 5: Error Landscape

("iso-layer-curves"). The graphs indicate that there is an optimal number $N_{opt}^{(1)}$ of nodes for which the performance of the single layer network has a best value $P_{opt}^{(1)}$. Within the single layer architecture, additional nodes lead to a decrease of performance due to overfitting. This can only be avoided if the training set is enlarged, since $N_{opt}^{(1)}$ grows with the number of available training examples.

However, Figs. 4 and 5 show that adding more units in the form of an additional, cascaded layer allows to increase performance *significantly beyond* $P_{opt}^{(1)}$. Similarly, the optimal performance of the resulting two-layer network cannot be improved beyond an optimal value $P_{opt}^{(2)}$ by arbitrarily increasing the number of nodes in the two-layer system. However, adding a third cascaded layer again allows to make use of more nodes to improve performance further, although this time the relative gain is smaller than for the first cascade step. The same situation repeats for larger numbers of cascaded layers. This suggests that the cascade architecture is very suitable to exploit the computational capabilities of large numbers of nodes for the task of building networks that *generalize well from small data sets without running into the problem of overfitting when many nodes are used.*

A second way of comparing the benefits of shallow and broad versus narrow and deep architectures is to compare the performance achieveable by distributing a fixed number $N$ of nodes over different numbers $L$ of cascaded layers. Fig. 6 shows the result for the same benchmark problem as in Fig. 4, each graph belonging to one of the values $N = 40, 60, 120, 240$ nodes and representing the NRMSE for distributing the $N$ nodes among $L$ layers of $N/L$ nodes each[1], $L$ ranging from 1 to 10 layers ("iso-nodes-curves").

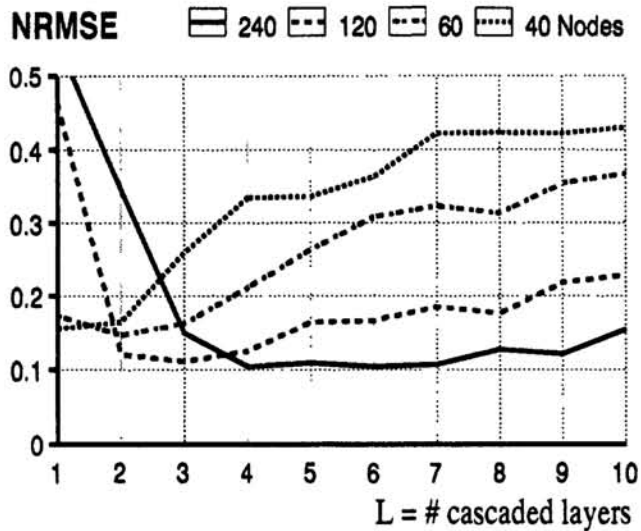

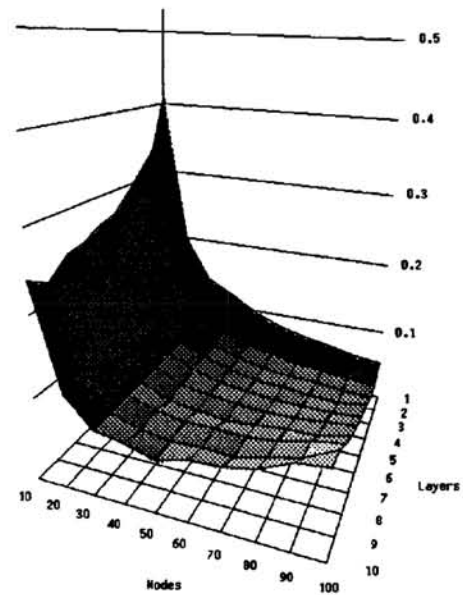

Figure 6: Iso-Nodes-Dependence          Figure 7: Nodes-Layer-Dependence

The results show that

(i) the optimal number of layers increases monotonously with — and is roughly proportional to — the number of nodes to be used.

(ii) if for each number of nodes the optimal number of layers is used, performance increases monotonously with the number of available nodes, and thus, as a consequence of (i), with the number of cascaded layers.

These results are not restricted to small data sets only. The application of the cascade algorithm is also useful if larger training sets are available. Fig. 7 represents the performance of LLM cascade networks on the test set after 300 training epochs overall on a learning set consisting of *5000 samples*. As could be expected, there is still no sign of overfitting, even using LLM networks with 100 nodes per layer. But regardless of the size of the single LLM unit, network performance is improved by the cascade process at least in a zone involving a total of some 300 nodes in the whole cascade.

## 5    Conclusions

Summarizing, we find that Cascade Network Architectures allow to use the benefits of large numbers of nodes even for small training data sets, and still bypass the problem of overfitting. To achieve this, the "width" of each layer must be matched to the size of the training set. The "depth" of the cascade then is determined by the total number of nodes available.

**Acknowledgements**

This work was supported by the German Ministry of Research and Technology (BMFT), Grant No. ITN9104AO. Any responsibility for the contents of this publication is with the authors.

## Footnotes

[1]rounding to the nearest integral, whenever $N/L$ is nonintegral.

# References

[1] Fahlman, S.E., and Lebiere, C. (1989), "The Cascade-Correlation Learning Architecture", in *Advances in Neural Information Processing Systems II*, ed. D.S. Touretzky, pp. 524–532.

[2] Kohonen, T. (1984), *Self-Organization and Associative Memory*, Springer Series in Information Sciences 8, Springer, Heidelberg.

[3] Kohonen, T. (1990), "The Self-Organizing Map", in *Proc. IEEE* **78**, pp. 1464–1480.

[4] Lapedes, A., and Farber, R. (1987), "Nonlinear signal processing using neural networks; Prediction and system modeling", TR LA–UR–87–2662

[5] Littmann, E., Ritter, H. (1992), "Cascade Network Architectures", in *Proc. Intern. Joint Conference On Neural Networks*, pp. II/398-404, Baltimore.

[6] Littmann, E., Ritter, H. (1992), "Cascade LLM Networks", in *Artificial Neural Networks II*, eds. I. Aleksander, J. Taylor, pp. 253-257, Elsevier Science Publishers (North Holland).

[7] Littmann, E., Meyering, A., Ritter, H. (1992), "Cascaded and Parallel Neural Network Architectures for Machine Vision — A Case Study", in *Proc. 14. DAGM-Symposium 1992, Dresden*, ed. S. Fuchs, pp. 81-87, Springer, Heidelberg.

[8] Mackey, M., and Glass, L. (1977), "Oscillations and chaos in physiological control systems", in *Science*, pp. 287–289.

[9] Moody, J., Darken, C. (1988). "Learning with Localized Receptive Fields", in *Proc. of the 1988 Connectionist Models Summer School*, Pittsburg, pp. 133–143, Morgan Kaufman Publishers, San Mateo, CA.

[10] Poggio, T., Edelman, S. (1990), "A network that learns to recognize three-dimensional objects", in *Nature* **343**, pp. 263–266.

[11] Ritter, H. (1991), "Learning with the Self-organizing Map", in *Artificial Neural Networks 1*, eds. T. Kohonen, K. Mäkisara, O. Simula, J. Kangas, pp. 357-364, Elsevier Science Publishers (North-Holland).

[12] Ritter, H., Martinetz, T., Schulten, K. (1992). *Neural Computation and Self-organizing Maps*, Addison-Wesley, Reading, MA.

[13] Walter, J., Ritter, H., Schulten, K. (1990). "Non-linear prediction with self-organizing maps", in *Proc. Intern. Joint Conference On Neural Networks*, San Diego, Vol.1, pp. 587–592.